# Exploiting Chaos to Control the Future

Gary W. Flake*      Guo-Zhen Sun[†]      Yee-Chun Lee[†]

Hsing-Hen Chen[†]

Institute for Advance Computer Studies
University of Maryland
College Park, MD   20742

## Abstract

Recently, Ott, Grebogi and Yorke (OGY) [6] found an effective
method to control chaotic systems to unstable fixed points by us-
ing only small control forces; however, OGY's method is based on
and limited to a linear theory and requires considerable knowledge
of the dynamics of the system to be controlled. In this paper we use
two radial basis function networks: one as a model of an unknown
plant and the other as the controller. The controller is trained
with a recurrent learning algorithm to minimize a novel objective
function such that the controller can locate an unstable fixed point
and drive the system into the fixed point with no *a priori* knowl-
edge of the system dynamics. Our results indicate that the neural
controller offers many advantages over OGY's technique.

## 1   Introduction

Recently, Ott, Grebogi and Yorke (OGY) [6] proposed a simple but very good idea.
Since any small perturbation can cause a large change in a chaotic trajectory, it
is possible to use a very small control force to achieve a large trajectory modifi-
cation. Moreover, due to the ergodicity of chaotic motion, any state in a chaotic

*Department of Computer Science, peyote@umiacs.umd.edu
[†]Laboratory for Plasma Research

attractor can be reached by a small control force. Since OGY published their work, several experiments and simulations have proven the usefulness of OGY's method. One prominent application of OGY's method is the prospect of controlling cardiac chaos [1].

We note that there are several unfavorable constraints on OGY's method. First, it requires *a priori* knowledge of the system dynamics, that is, the location of fixed points. Second, due to the limitation of linear theory, it will not work in the presence of large noise or when the control force is as large as beyond the linear region from which the control law was constructed. Third, although the ergodicity theory guarantees that any state after moving away from the desired fixed point will eventually return to its linear vicinity, it may take a very long time for this to happen, especially for a high dimensional chaotic attractor.

In this paper we will demonstrate how a neural network (NN) can control a chaotic system with only a small control force and be trained with only examples from the state-space. To solve this problem, we introduced a novel objective function which measures the distance between the current state and its previous average. By minimizing this objective function, the NN can automatically locate the fixed point. As a preliminary step, a training set is used to train a forward model for the chaotic dynamics. The work of Jordan and Rumelhart [4] has shown that control problems can be mapped into supervised learning problems by coupling the outputs of a controller NN (the control signals) to the inputs of a forward model of a plant to form a multilayer network that is indirectly recurrent. A recurrent learning algorithm is used to train the controller NN. To facilitate learning we use an extended radial basis function (RBF) network for both the forward model and the controller. To benchmark with OGY's result, the Hènon map is used as a numerical example. The numerical results have shown the preliminary success of the proposed scheme. Details will be given in the following sections.

In the next section we give our methodology and describe the general form of the recurrent learning algorithm used in our experiments. In Section 3, we discuss RBF networks and reintroduce a more powerful version. In Section 4, the numerical results are presented in detail. Finally, in Section 5, we give our conclusions.

## 2    Recurrent Learning for Control

Let $\vec{k}(\cdot)$ denote a NN whose output, $\vec{u}_t$, is composed through a plant, $\vec{f}(\cdot)$, with unknown dynamics. The output of the unknown plant (the state), $\vec{x}_{t+1}$, forms part of the input for the NN at the next time step, hence the recurrency. At each time step the state is also passed to an output function, $\vec{g}(\cdot)$, which computes the sensation, $\vec{y}_{t+1}$. The time evolution of this system is more accurately described by

$$
\begin{aligned}
\vec{u}_t &= \vec{k}(\vec{x}_t, \vec{y}_{t+1}^*, \vec{w}) \\
\vec{x}_{t+1} &= \vec{f}(\vec{x}_t, \vec{u}_t) \\
\vec{y}_{t+1} &= \vec{g}(\vec{x}_{t+1}),
\end{aligned}
$$

where $\vec{y}_{t+1}^*$ is the desired sensation for time $t+1$ and $\vec{w}$ represents the trainable weights for the network. Additionally, we define the temporally local and global

error functionals

$$J_t = \tfrac{1}{2}\|\vec{y_t^*} - \vec{y_t}\|^2 \quad \text{and} \quad E = \sum_{i=1}^{N} J_i,$$

where $N$ is the final time step for the system.

The real-time recurrent learning (RTRL) algorithm [9] for training the network weights to minimize $E$ is based on the fair assumption that minimizing the local error functionals with a small learning rate at each time step will correspond to minimizing the global error. To derive the learning algorithm, we can imagine the system consisting of the plant, controller, and error functionals as being unfolded in time. From this perspective we can view each instance of the controller NN as a separate NN and thus differentiate the error functionals with respect to the network weights at different times. Hence, we now add a time index to $\vec{w_t}$ to represent this fact. However, when we use $\vec{w}$ without the time index, the term should be understood to be time invariant.

We can now define the matrix

$$\Gamma_t = \sum_{i=0}^{t-1} \frac{\partial \vec{x_t}}{\partial \vec{w_i}} = \frac{\partial \vec{x_t}}{\partial \vec{u}_{t-1}} \frac{\partial \vec{u}_{t-1}}{\partial \vec{w}_{t-1}} + \left( \frac{\partial \vec{x_t}}{\partial \vec{u}_{t-1}} \frac{\partial \vec{u}_{t-1}}{\partial \vec{x}_{t-1}} + \frac{\partial \vec{x_t}}{\partial \vec{x}_{t-1}} \right) \Gamma_{t-1}, \tag{1}$$

which further allows us to define

$$\frac{\partial J_i}{\partial \vec{w}} = \frac{\partial J_i}{\partial \vec{y_i}} \frac{\partial \vec{y_i}}{\partial \vec{x_i}} \Gamma_i \tag{2}$$

$$\frac{\partial E}{\partial \vec{w}} = \sum_{i=1}^{N} \frac{\partial J_i}{\partial \vec{w}}. \tag{3}$$

Equation 2 is the gradient equation for the RTRL algorithm while Equation 3 is for the backpropagation through time (BPTT) learning algorithm [7]. The gradients defined by these equations are usually used with gradient descent on a multilayer perceptron (MLP). We will use them on RBF networks.

## 3  The CNLS Network

The Connectionist Normalized Local Spline (CNLS) network [3] is an extension of the more familiar radial basis function network of Moody and Darken [5]. The forward operation of the network is defined by

$$\phi(\vec{x}) = \sum_i (f_i + \vec{d_i} \cdot (\vec{x} - \vec{a_i})) \, \rho_i(\vec{x}), \tag{4}$$

where

$$\rho_i(\vec{x}) = \frac{\exp(-\beta_i\|\vec{x} - \vec{a_i}\|^2)}{\sum_j \exp(-\beta_j\|\vec{x} - \vec{a_j}\|^2)}. \tag{5}$$

All of the equations in this section assume a single output. Generalizing them for multiple outputs merely adds another index to the terms. For all of our simulations, we choose to distribute the centers, $\vec{a_i}$, based on a sample of the input space.

Additionally, the basis widths, $\beta_i$, are set to an experimentally determined constant. Because the output, $\phi$, is linear in the terms $f_i$ and $\vec{d}_i$, training them is very fast. To train the CNLS network on a prediction problem we, can use a quadratic error function of the form $E = \frac{1}{2}(y(\vec{x}) - \phi(\vec{x}))^2$, where $y(\vec{x})$ is the target function that we wish to approximate. We use a one-dimensional Newton-like method [8] which yields the update equations

$$f_i^{t+1} \;=\; f_i^t - E\,\frac{\frac{\partial E}{\partial f_i^t}}{\sum_k \frac{\partial E}{\partial f_k^t}^2} \;=\; f_i^t + \eta\,(y(\vec{x}) - \phi(\vec{x}))\frac{\rho_i(\vec{x})}{\sum_k \rho_k^2(\vec{x})},$$

$$\vec{d}_i^{t+1} \;=\; \vec{d}_i^t - E\,\frac{\frac{\partial E}{\partial d_i^t}}{\sum_k \|\frac{\partial E}{\partial d_k^t}\|^2} \;=\; \vec{d}_i^t + \eta\,(y(\vec{x}) - \phi(\vec{x}))\frac{(\vec{x} - \vec{a}_i)\,\rho_i(\vec{x})}{\sum_k \|\vec{x} - \vec{a}_k\|^2 \rho_k^2(\vec{x})}.$$

The right-most update rules form the learning algorithm when using the CNLS network for prediction, where $\eta$ is a learning rate that should be set below 1.0. The left-most update rules describe a more general learning algorithm that can be used when a target output is unknown.

When using the CNLS network architecture as part of a recurrent learning algorithm we must be able to differentiate the network outputs with respect to the inputs. Note that in Equations 1 and 2 each of the terms $\partial \vec{x}_t / \partial \vec{u}_{t-1}$, $\partial \vec{u}_{t-1} / \partial \vec{x}_{t-1}$, $\partial \vec{x}_t / \partial \vec{x}_{t-1}$, and $\partial \vec{y}_i / \partial \vec{x}_i$ can either be exactly solved or approximated by differentiating a CNLS network. Since the CNLS output is highly nonlinear in its inputs, computing these partial derivatives is not quite as elegant as it would be in a MLP. Nevertheless, it can be done. We skip the details and just show the end result:

$$\frac{\partial \phi}{\partial \vec{x}} = \sum_{i=1}^{n} \vec{d}_i \rho_i(\vec{x}) + 2\sum_{j=1}^{n}(\rho_j(\vec{x})\,q_j\,\beta_j\,(\vec{a}_j - \vec{x})) - 2\phi(\vec{x})\vec{\gamma}, \tag{6}$$

with $\vec{\gamma} = \sum_k \rho_k(\vec{x})\beta_k(\vec{a}_k - \vec{x})$, and $q_i = f_i + \vec{d}_i \cdot (\vec{x} - \vec{a}_i)$.

## 4  Adaptive Control

By combining the equations from the last two sections, we can construct a recurrent learning scheme for RBF networks in a similar fashion to what has been done with MLP networks. To demonstrate the utility of our technique, we have chosen a well-studied nonlinear plant that has been successfully modeled and controlled by using non-neural techniques. Specifically, we will use the Hènon map as a plant, which has been the focus of much of the research of OGY [6]. We also adopt some of their notation and experimental constraints.

### 4.1  The Hènon Map

The Hènon map [2] is described by the equations

$$x_{t+1} \;=\; A - x_t^2 + By_t \tag{7}$$

$$y_{t+1} \;=\; x_t, \tag{8}$$

where $A = A_0 + p$ and $p$ is a control parameter that may be modified at each time step to coerce the plant into a desirable state. For all simulations we set $A_0 = 1.29$ and $B = 0.3$ which gives the above equations a chaotic attracter that also contains an unstable fixed point. Our goal is to train a CNLS network that can locate and drive the map into the unstable fixed point and keep it there with only a minimal amount of information about the plant and by using only small values of $p$.

The unstable fixed point $(x_F, y_F)$ in Equations 7 and 8 can be easily calculated as $x_F = y_F \approx 0.838486$. Forcing the Hènon map to the fixed point is trivial if the controller is given unlimited control of the parameter. To make the problem more realistic we define $p^*$ as the maximum magnitude that $p$ can take and use the rule below on the left

$$p_n = \begin{cases} p & \text{if } |p| < p^* \\ p^* & \text{if } p > p^* \\ -p^* & \text{if } p < -p^* \end{cases} \qquad\qquad p_n = \begin{cases} p & \text{if } |p| < p^* \\ 0 & \text{if } |p| > p^* \end{cases}$$

while OGY use the rule on the right. The reason we avoid the second rule is that it cannot be modeled by a CNLS network with any precision since it is step-like.

The next task is to define what it means to "control" the Hènon map. Having analytical knowledge of the fixed point in the attracter would make the job of the controller much easier, but this is unrealistic in the case where the dynamics of the plant to control are unknown. Instead, we use an error function that simply compares the current state of the plant with an average of previous states:

$$e_t = \frac{1}{2}\left[ (x_t - \langle x \rangle_\tau)^2 + (y_t - \langle y \rangle_\tau)^2 \right], \tag{9}$$

where $\langle \cdot \rangle_\tau$ is the average of the last $\tau$ values of its argument. This function approaches zero when the map is in a fixed point for time length greater than $\tau$. This function requires no special knowledge about the dynamics of the plant, yet it still enforces our constraint of driving the map into a fixed point.

The learning algorithm also requires the partial derivatives of the error function with respect to the plant state variables, which are $\partial e_t / \partial x_t = x_t - \langle x \rangle_\tau$ and $\partial e_t / \partial y_t = y_t - \langle y \rangle_\tau$. These two equations and the objective function are the only special purpose equations used for this problem. All other equations generalize from the derivation of the algorithm. Additionally, since the "output" representation (as discussed earlier) is identical to the state representation, training on a distinct output function is not strictly necessary in this case. Thus, we simplify the problem by only using a single additional model for the unknown next-state function of the Hènon map.

## 4.2  Simulation

To facilitate comparison between alternate control techniques, we now introduce the term $\epsilon \vec{\delta}_t$ where $\vec{\delta}_t$ is a random variable and $\epsilon$ is a small constant which specifies the intensity of the noise. We use a Gaussian distribution for $\vec{\delta}_t$ such that the distribution has a zero mean, is independent, and has a variance of one. In keeping with [6], we discard any values of $\vec{\delta}_t$ which are greater in magnitude than 10. For training we set $\epsilon = 0.038$. However, for tests on the real controller, we will show results for several values of $\epsilon$.

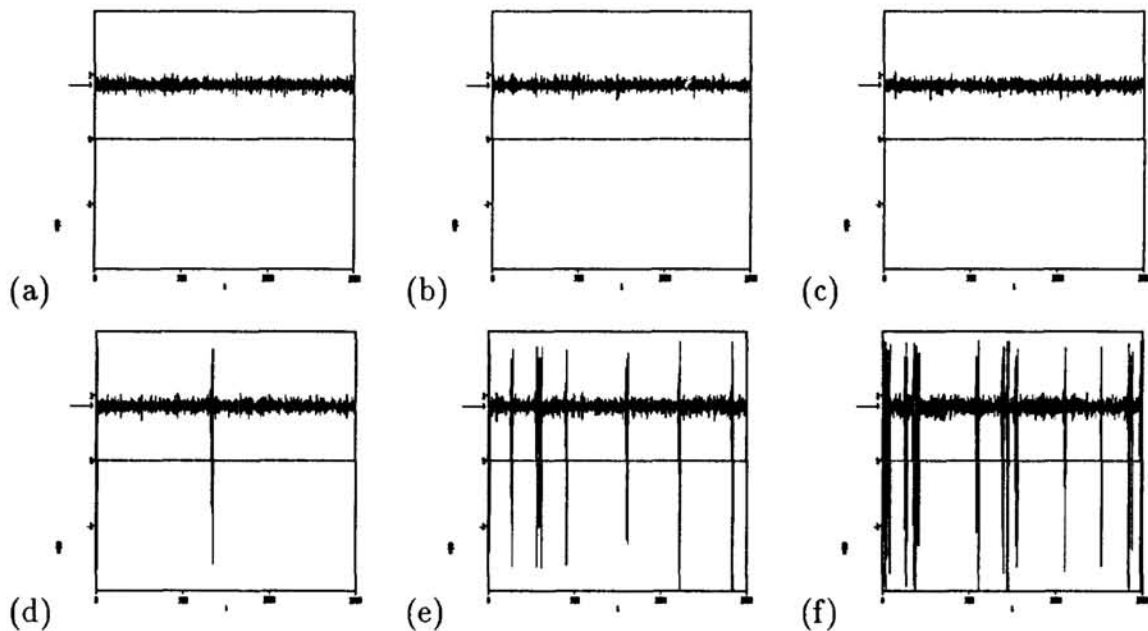

Figure 1: Experimental results from training a neural controller to drive the Hènon map into a fixed point. From (a) to (f), the values of $\epsilon$ are 0.035, 0.036, 0.038, 0.04, 0.05, and 0.06, respectively. The top row corresponds to identical experiments performed in [6].

We add the noise in two places. First, when training the model, we add noise to the target output of the model (the next state). Second, when testing the controller on the real Hènon map, we add the noise to the input of the plant (the previous state). In the second case, we consider the noise to be an artifact of our fictional measurements; that is, the plant evolves from the previous noise free state.

Training the controller is done in two stages: an off-line portion to tune the model and an on-line stage to tune the controller. To train the model we randomly pick a starting state within a region $(-1.5, 1.5)$ for the two state variables. We then iterate the map for one hundred cycles with $p = 0$ so that the points will converge onto the chaotic attractor. Next, we randomly pick a value for $p$ in the range of $(-p^*, p^*)$. The last state from the iteration is combined with this control parameter to compute a target state. We then add the noise to the new state values. Thus, the model input consists of a clean previous state and a control parameter and the target values consist of the noisy next state. We compute 100 training patterns in this manner. Using the prediction learning algorithm for the CNLS network we train the model network on each of the 100 patterns (in random order) for 30 epochs. The model quickly converges to a low average error.

In the next stage, we use the model network to train the controller network in two ways. First, the model acts as the plant for the purposes of computing a next state. Additionally, we differentiate the model for values needed for the RTRL algorithm. We train the controller for 30 epochs, where each epoch consists of 50 cycles. At the beginning of each epoch we initialize the plant state to some random values (not necessarily on the chaotic attracter,) and set the recurrent history matrix,

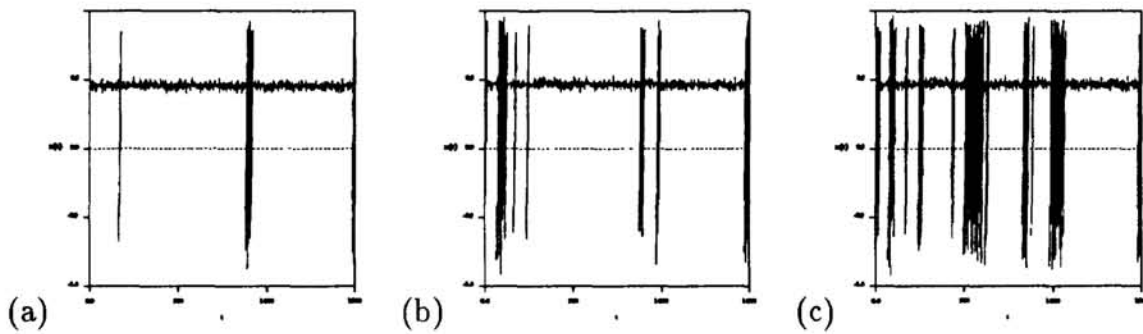

Figure 2: Experimental results from [6]. From left to right, the values of $\epsilon$ are 0.035, 0.036, and 0.038, respectively.

$\Gamma_t$, to zero. Then, for each cycle, we feed the previous state into the controller as input. This produces a control parameter which is fed along with the previous state as input into the model network, which in turn produces the next state. This next state is fed into the error function to produce the error signal. At this point we compute all of the necessary values to train the controller for that cycle while maintaining the history matrix.

In this way, we train both the model and control networks with only 100 data points, since the controller never sees any of the real values from the Hènon map but only estimates from the model. For this experiment both the control and model RBF networks consist of 40 basis functions.

### 4.3   Summary

Our results are summarized by Figure 1. As can be seen, the controller is able to drive the Hènon Map into the fixed point very rapidly and it is capable of keeping it there for an extended period of time without transients. As the level of noise is increased, it can be seen that the plant maintains control for quite some time. The first visible spike can be observed when $\epsilon = 0.04$.

These results are an improvement over the results generated from the best non-neural technique available for two reasons: First, the neural controller that we have trained is capable of driving the Hènon map into a fixed point with far fewer transients then other techniques. Specifically, alternate techniques, as illustrated in Figure 2, experience numerous spikes in the map for values of $\epsilon$ for which our controller is spike-free (0.035 − 0.038). Second, our training technique has smaller data requirements and uses less special purpose information. For example, the RBF controller was trained with only 100 data points compared to 500 for the non-neural. Additionally, non-neural techniques will typically estimate the location of the fixed point with an initial data set. In the case of [6] it was assumed that the fixed point could be easily discovered by some technique, and as a result all of their experiments rely on the true (hard-coded) fixed point. This, of course, could be discovered by searching the input space on the RBF model, but we have instead allowed the controller to discover this feature on its own.

## 5    Conclusion and Future Directions

A crucial component of the success of our approach is the objective function that measures the distance between the current state and the nearest time average. The reason why this objective function works is that during the control stage the learning algorithm is minimizing only a small distance between the current point and the "moving target." This is in contrast to minimizing the large distance between the current point and the target point, which usually causes unstable long time correlation in chaotic systems and ruins the learning. The carefully designed recurrent learning algorithm and the extended RBF network also contribute to the success of this approach. Our results seem to indicate that RBF networks hold great promise in recurrent systems. However, further study must be done to understand why and how NNs could provide more useful schemes to control real world chaos.

**Acknowledgements**

We gratefully acknowledge helpful comments from and discussions with Chris Barnes, Lee Giles, Roger Jones, Ed Ott, and James Reggia. This research was supported in part by AFOSR grant number F49620-92-J-0519.

# References

[1] A. Garfinkel, M.L. Spano, and W.L. Ditto. Controlling cardiac chaos. *Science*, 257(5074):1230, August 1992.

[2] M. Hènon. A two-dimensional mapping with a strange attractor. *Communications in Mathematical Physics*, 50:69–77, 1976.

[3] R.D. Jones, Y.C. Lee, C.W. Barnes, G.W. Flake, K. Lee, P.S. Lewis, and S. Qian. Function approximation and time series prediction with neural network. In *Proceedings of the International Joint Conference on Neural Networks*, 1990.

[4] M.I. Jordan and D.E. Rumelhart. Forward models: Supervised learning with a distal teacher. Technical Report Occasional Paper #40, MIT Center for Cognitive Science, 1990.

[5] J. Moody and C. Darken. Fast learning in networks of locally-tuned processing units. *Neural Computation*, 1:281–294, 1989.

[6] E. Ott, C. Grebogi, and J.A. Yorke. Controlling chaotic dynamical systems. In D.K. Campbell, editor, *CHAOS: Soviet-American Perspectives on Nonlinear Science*, pages 153–172. American Institute of Physics, New York, 1990.

[7] F.J. Pineda. Generalization of back-propagation to recurrent neural networks. *Physical Review Letters*, 59:2229–2232, 1987.

[8] W.H. Press, B.P. Flannery, S.A. Teukolsky, and W.T. Vetterling. *Numerical Recipes*. Cambridge University Press, Cambridge, 1986.

[9] R.J. Williams and D. Zipser. Experimental analysis of the real-time recurrent learning algorithm. *Connection Science*, 1:87–111, 1989.